# Efficient Direct Density Ratio Estimation for Non-stationarity Adaptation and Outlier Detection

**Takafumi Kanamori**
Nagoya University
Nagoya, Japan
kanamori@is.nagoya-u.ac.jp

**Shohei Hido**
IBM Research
Kanagawa, Japan
hido@jp.ibm.com

**Masashi Sugiyama**
Tokyo Institute of Technology
Tokyo, Japan
sugi@cs.titech.ac.jp

## Abstract

We address the problem of estimating the ratio of two probability density functions (a.k.a. the *importance*). The importance values can be used for various succeeding tasks such as *non-stationarity adaptation* or *outlier detection*. In this paper, we propose a new importance estimation method that has a closed-form solution; the leave-one-out cross-validation score can also be computed analytically. Therefore, the proposed method is computationally very efficient and numerically stable. We also elucidate theoretical properties of the proposed method such as the convergence rate and approximation error bound. Numerical experiments show that the proposed method is comparable to the best existing method in accuracy, while it is computationally more efficient than competing approaches.

## 1 Introduction

In the context of *importance sampling*, the ratio of two probability density functions is called the *importance*. The problem of estimating the importance is gathering a lot of attention these days since the importance can be used for various succeeding tasks, e.g.,

*Covariate shift adaptation:* Covariate shift is a situation in supervised learning where the distributions of inputs change between the training and test phases but the conditional distribution of outputs given inputs remains unchanged [8]. Covariate shift is conceivable in many real-world applications such as bioinformatics, brain-computer interfaces, robot control, spam filtering, and econometrics. Under covariate shift, standard learning techniques such as maximum likelihood estimation or cross-validation are biased and therefore unreliable—the bias caused by covariate shift can be compensated by weighting the training samples according to the importance [8, 5, 1, 9].

*Outlier detection:* The outlier detection task addressed here is to identify irregular samples in an evaluation dataset based on a model dataset that only contains regular samples [7, 3]. The importance values for regular samples are close to one, while those for outliers tend to be significantly deviated from one. Thus the values of the importance could be used as an index of the degree of outlyingness.

Below, we refer to the two sets of samples as the training and test sets. A naive approach to estimating the importance is to first estimate the training and test densities from the sets of training and test samples separately, and then take the ratio of the estimated densities. However, density estimation is known to be a hard problem particularly in high-dimensional cases. In practice, such an appropriate parametric model may not be available and therefore this naive approach is not so effective.

To cope with this problem, we propose a direct importance estimation method that does not involve density estimation. The proposed method, which we call *least-squares importance fitting* (LSIF), is formulated as a convex quadratic program and therefore the unique global solution can be obtained. We give a cross-validation method for model selection and a regularization path tracking algorithm for efficient computation [4].

This regularization path tracking algorithm is turned out to be computationally very efficient since the entire solution path can be traced without a quadratic program solver. However, it tends to share a common weakness of path tracking algorithms, i.e., *accumulation of numerical errors*. To overcome this drawback, we develop an approximation algorithm called *unconstrained LSIF* (uLSIF), which allows us to obtain the closed-form solution that can be stably computed just by solving a system of linear equations. Thus uLSIF is computationally efficient and numerically stable. Moreover, the leave-one-out error of uLSIF can also be computed analytically, which further improves the computational efficiency in model selection scenarios.

We experimentally show that the accuracy of uLSIF is comparable to the best existing method while its computation is much faster than the others in covariate shift adaptation and outlier detection.

## 2   Direct Importance Estimation

**Formulation and Notation:**   Let $\mathcal{D} \subset (\mathbb{R}^d)$ be the data domain and suppose we are given independent and identically distributed (i.i.d.) training samples $\{\boldsymbol{x}_i^{\mathrm{tr}}\}_{i=1}^{n_{\mathrm{tr}}}$ from a training distribution with density $p_{\mathrm{tr}}(\boldsymbol{x})$ and i.i.d. test samples $\{\boldsymbol{x}_j^{\mathrm{te}}\}_{j=1}^{n_{\mathrm{te}}}$ from a test distribution with density $p_{\mathrm{te}}(\boldsymbol{x})$. We assume $p_{\mathrm{tr}}(\boldsymbol{x}) > 0$ for all $\boldsymbol{x} \in \mathcal{D}$. The goal of this paper is to estimate the *importance*

$$w(\boldsymbol{x}) = \frac{p_{\mathrm{te}}(\boldsymbol{x})}{p_{\mathrm{tr}}(\boldsymbol{x})}$$

from $\{\boldsymbol{x}_i^{\mathrm{tr}}\}_{i=1}^{n_{\mathrm{tr}}}$ and $\{\boldsymbol{x}_j^{\mathrm{te}}\}_{j=1}^{n_{\mathrm{te}}}$. Our key restriction is that we want to avoid estimating densities $p_{\mathrm{te}}(\boldsymbol{x})$ and $p_{\mathrm{tr}}(\boldsymbol{x})$ when estimating the importance $w(\boldsymbol{x})$.

**Least-squares Approach:**   Let us model the importance $w(\boldsymbol{x})$ by the following linear model:

$$\widehat{w}(\boldsymbol{x}) = \boldsymbol{\alpha}^\top \boldsymbol{\varphi}(\boldsymbol{x}), \tag{1}$$

where $^\top$ denotes the transpose, $\boldsymbol{\alpha} = (\alpha_1, \ldots, \alpha_b)^\top$, is a parameter to be learned, $b$ is the number of parameters, $\boldsymbol{\varphi}(\boldsymbol{x}) = (\varphi_1(\boldsymbol{x}), \ldots, \varphi_b(\boldsymbol{x}))^\top$ are basis functions such that $\boldsymbol{\varphi}(\boldsymbol{x}) \geq \mathbf{0}_b$ for all $\boldsymbol{x} \in \mathcal{D}$, $\mathbf{0}_b$ denotes the $b$-dimensional vector with all zeros, and the inequality for vectors is applied in the element-wise manner. Note that $b$ and $\{\varphi_\ell(\boldsymbol{x})\}_{\ell=1}^b$ could be dependent on the samples i.e., *kernel models* are also allowed. We explain how the basis functions $\{\varphi_\ell(\boldsymbol{x})\}_{\ell=1}^b$ are chosen later.

We determine the parameter $\boldsymbol{\alpha}$ so that the following squared error is minimized:

$$J_0(\boldsymbol{\alpha}) = \tfrac{1}{2} \int \left( \widehat{w}(\boldsymbol{x}) - \frac{p_{\mathrm{te}}(\boldsymbol{x})}{p_{\mathrm{tr}}(\boldsymbol{x})} \right)^2 p_{\mathrm{tr}}(\boldsymbol{x}) d\boldsymbol{x} = \tfrac{1}{2} \int \widehat{w}(\boldsymbol{x})^2 p_{\mathrm{tr}}(\boldsymbol{x}) d\boldsymbol{x} - \int \widehat{w}(\boldsymbol{x}) p_{\mathrm{te}}(\boldsymbol{x}) d\boldsymbol{x} + C,$$

where $C = \tfrac{1}{2} \int w(\boldsymbol{x}) p_{\mathrm{te}}(\boldsymbol{x}) d\boldsymbol{x}$ is a constant and therefore can be safely ignored. Let

$$J(\boldsymbol{\alpha}) = J_0(\boldsymbol{\alpha}) - C = \tfrac{1}{2} \boldsymbol{\alpha}^\top \boldsymbol{H} \boldsymbol{\alpha} - \boldsymbol{h}^\top \boldsymbol{\alpha}, \tag{2}$$

where $\boldsymbol{H} = \int \boldsymbol{\varphi}(\boldsymbol{x}) \boldsymbol{\varphi}(\boldsymbol{x})^\top p_{\mathrm{tr}}(\boldsymbol{x}) d\boldsymbol{x}, \boldsymbol{h} = \int \boldsymbol{\varphi}(\boldsymbol{x}) p_{\mathrm{te}}(\boldsymbol{x}) d\boldsymbol{x}$. Using the empirical approximation and taking into account the non-negativity of the importance function $w(\boldsymbol{x})$, we obtain

$$\min_{\boldsymbol{\alpha} \in \mathbb{R}^b} \left[ \tfrac{1}{2} \boldsymbol{\alpha}^\top \widehat{\boldsymbol{H}} \boldsymbol{\alpha} - \widehat{\boldsymbol{h}}^\top \boldsymbol{\alpha} + \lambda \mathbf{1}_b^\top \boldsymbol{\alpha} \right] \quad \text{s.t. } \boldsymbol{\alpha} \geq \mathbf{0}_b, \tag{3}$$

where $\widehat{\boldsymbol{H}} = \frac{1}{n_{\mathrm{tr}}} \sum_{i=1}^{n_{\mathrm{tr}}} \boldsymbol{\varphi}(\boldsymbol{x}_i^{\mathrm{tr}}) \boldsymbol{\varphi}(\boldsymbol{x}_i^{\mathrm{tr}})^\top, \quad \widehat{\boldsymbol{h}} = \frac{1}{n_{\mathrm{te}}} \sum_{j=1}^{n_{\mathrm{te}}} \boldsymbol{\varphi}(\boldsymbol{x}_j^{\mathrm{te}})$. $\lambda \mathbf{1}_b^\top \boldsymbol{\alpha}$ is a regularization term for avoiding overfitting, $\lambda \geq 0$, and $\mathbf{1}_b$ is the $b$-dimensional vector with all ones.

The above problem is a convex quadratic program and therefore the global optimal solution can be obtained by a standard software. We call this method *Least-Squares Importance Fitting* (LSIF).

**Convergence Analysis of LSIF:** Here, we theoretically analyze the convergence property of the solution $\widehat{\boldsymbol{\alpha}}$ of the LSIF algorithm. Let $\boldsymbol{\alpha}^*$ be the optimal solution of the 'ideal' problem:

$$\min_{\boldsymbol{\alpha} \in \mathbb{R}^b} \left[ \tfrac{1}{2} \boldsymbol{\alpha}^\top \boldsymbol{H} \boldsymbol{\alpha} - \boldsymbol{h}^\top \boldsymbol{\alpha} + \lambda \mathbf{1}_b^\top \boldsymbol{\alpha} \right] \quad \text{s.t. } \boldsymbol{\alpha} \geq \mathbf{0}_b. \tag{4}$$

Let $f(n) = \omega(g(n))$ mean that $f(n)$ asymptotically dominates $g(n)$, i.e., for all $C > 0$, there exists $n_0$ such that $|Cg(n)| < |f(n)|$ for all $n > n_0$. Then we have the following theorem.

**Theorem 1** *Assume that (a) the optimal solution of the problem* (4) *satisfies the strict complementarity condition, and (b) $n_{\mathrm{tr}}$ and $n_{\mathrm{te}}$ satisfy $n_{\mathrm{te}} = \omega(n_{\mathrm{tr}}^2)$. Then we have $\mathbb{E}[J(\widehat{\boldsymbol{\alpha}})] = J(\boldsymbol{\alpha}^*) + \mathcal{O}\left(n_{\mathrm{tr}}^{-1}\right)$, where $\mathbb{E}$ denotes the expectation over all possible training samples of size $n_{\mathrm{tr}}$ and all possible test samples of size $n_{\mathrm{te}}$.*

Theorem 1 guarantees that LSIF converges to the ideal solution with order $n_{\mathrm{tr}}^{-1}$. It is possible to explicitly obtain the coefficient of the term of order $n_{\mathrm{tr}}^{-1}$, but we omit the detail due to lack of space.

**Model Selection for LSIF:** The performance of LSIF depends on the choice of the regularization parameter $\lambda$ and basis functions $\{\varphi_\ell(\boldsymbol{x})\}_{\ell=1}^b$ (which we refer to as a *model*). Since our objective is to minimize the cost function $J$, it is natural to determine the model such that $J$ is minimized.

Here, we employ cross-validation for estimating $J(\widehat{\boldsymbol{\alpha}})$, which has an accuracy guarantee for finite samples: First, the training samples $\{\boldsymbol{x}_i^{\mathrm{tr}}\}_{i=1}^{n_{\mathrm{tr}}}$ and test samples $\{\boldsymbol{x}_j^{\mathrm{te}}\}_{j=1}^{n_{\mathrm{te}}}$ are divided into $R$ disjoint subsets $\{\mathcal{X}_r^{\mathrm{tr}}\}_{r=1}^R$ and $\{\mathcal{X}_r^{\mathrm{te}}\}_{r=1}^R$, respectively. Then an importance estimate $\widehat{w}_r(\boldsymbol{x})$ is obtained using $\{\mathcal{X}_j^{\mathrm{tr}}\}_{j \neq r}$ and $\{\mathcal{X}_j^{\mathrm{te}}\}_{j \neq r}$, and the cost $J$ is approximated using the held-out samples $\mathcal{X}_r^{\mathrm{tr}}$ and $\mathcal{X}_r^{\mathrm{te}}$ as $\widehat{J}_r^{(\mathrm{CV})} = \frac{1}{2|\mathcal{X}_r^{\mathrm{tr}}|} \sum_{\boldsymbol{x}^{\mathrm{tr}} \in \mathcal{X}_r^{\mathrm{tr}}} \widehat{w}_r(\boldsymbol{x}^{\mathrm{tr}})^2 - \frac{1}{|\mathcal{X}_r^{\mathrm{te}}|} \sum_{\boldsymbol{x}^{\mathrm{te}} \in \mathcal{X}_r^{\mathrm{te}}} \widehat{w}_r(\boldsymbol{x}^{\mathrm{te}})$. This procedure is repeated for $r = 1, \ldots, R$ and its average $\widehat{J}^{(\mathrm{CV})}$ is used as an estimate of $J$. We can show that $\widehat{J}^{(\mathrm{CV})}$ gives an almost unbiased estimate of the true cost $J$, where the 'almost'-ness comes from the fact that the number of samples is reduced due to data splitting.

**Heuristics of Basis Function Design:** A good model may be chosen by cross-validation, given that a family of promising model candidates is prepared. As model candidates, we propose using a Gaussian kernel model centered at the *test* input points $\{\boldsymbol{x}_j^{\mathrm{te}}\}_{j=1}^{n_{\mathrm{te}}}$, i.e.,

$$\widehat{w}(\boldsymbol{x}) = \sum_{\ell=1}^{n_{\mathrm{te}}} \alpha_\ell K_\sigma(\boldsymbol{x}, \boldsymbol{x}_\ell^{\mathrm{te}}), \quad \text{where} \quad K_\sigma(\boldsymbol{x}, \boldsymbol{x}') = \exp\left(-\|\boldsymbol{x} - \boldsymbol{x}'\|^2/(2\sigma^2)\right). \tag{5}$$

The reason why we chose the test input points $\{\boldsymbol{x}_j^{\mathrm{te}}\}_{j=1}^{n_{\mathrm{te}}}$ as the Gaussian centers, not the training input points $\{\boldsymbol{x}_i^{\mathrm{tr}}\}_{i=1}^{n_{\mathrm{tr}}}$, is as follows. By definition, the importance $w(\boldsymbol{x})$ tends to take large values if the training input density $p_{\mathrm{tr}}(\boldsymbol{x})$ is small and the test input density $p_{\mathrm{te}}(\boldsymbol{x})$ is large; conversely, $w(\boldsymbol{x})$ tends to be small (i.e., close to zero) if $p_{\mathrm{tr}}(\boldsymbol{x})$ is large and $p_{\mathrm{te}}(\boldsymbol{x})$ is small. When a function is approximated by a Gaussian kernel model, many kernels may be needed in the region where the output of the target function is large; on the other hand, only a small number of kernels would be enough in the region where the output of the target function is close to zero. Following this heuristic, we allocate many kernels at high *test* input density regions, which can be achieved by setting the Gaussian centers at the test input points $\{\boldsymbol{x}_j^{\mathrm{te}}\}_{j=1}^{n_{\mathrm{te}}}$.

Alternatively, we may locate $(n_{\mathrm{tr}} + n_{\mathrm{te}})$ Gaussian kernels at both $\{\boldsymbol{x}_i^{\mathrm{tr}}\}_{i=1}^{n_{\mathrm{tr}}}$ and $\{\boldsymbol{x}_j^{\mathrm{te}}\}_{j=1}^{n_{\mathrm{te}}}$. However, in our preliminary experiments, this did not further improve the performance, but just slightly increased the computational cost. When $n_{\mathrm{te}}$ is large, just using all the test input points $\{\boldsymbol{x}_j^{\mathrm{te}}\}_{j=1}^{n_{\mathrm{te}}}$ as Gaussian centers is already computationally rather demanding. To ease this problem, we practically propose using a subset of $\{\boldsymbol{x}_j^{\mathrm{te}}\}_{j=1}^{n_{\mathrm{te}}}$ as Gaussian centers for computational efficiency, i.e.,

$$\widehat{w}(\boldsymbol{x}) = \sum_{\ell=1}^b \alpha_\ell K_\sigma(\boldsymbol{x}, \boldsymbol{c}_\ell), \tag{6}$$

where $\boldsymbol{c}_\ell$ is a template point randomly chosen from $\{\boldsymbol{x}_j^{\mathrm{te}}\}_{j=1}^{n_{\mathrm{te}}}$ and $b \ (\leq n_{\mathrm{te}})$ is a prefixed number. In the experiments shown later, we fix the number of template points at $b = \min(100, n_{\mathrm{te}})$, and optimize the kernel width $\sigma$ and the regularization parameter $\lambda$ by cross-validation with grid search.

**Entire Regularization Path for LSIF:** We can show that the LSIF solution $\widehat{\boldsymbol{\alpha}}$ is piecewise linear with respect to the regularization parameter $\lambda$. Therefore, the *regularization path* (i.e., solutions for all $\lambda$) can be computed efficiently based on the *parametric optimization technique* [4].

A basic idea of regularization path tracking is to check the violation of the Karush-Kuhn-Tucker (KKT) conditions—which are necessary and sufficient conditions for optimality of convex programs—when the regularization parameter $\lambda$ is changed. Although the detail of the algorithm is omitted due to lack of space, we can show that a quadratic programming solver is no longer needed for obtaining the entire solution path of LSIF—just computing matrix inverses is enough. This highly contributes to saving the computation time. However, in our preliminary experiments, the regularization path tracking algorithm is turned out to be numerically rather unreliable since the numerical errors tend to be accumulated when tracking the regularization path. This seems to be a common pitfall of solution path tracking algorithms in general.

## 3 Approximation Algorithm

**Unconstrained Least-squares Approach:** The approximation idea we introduce here is very simple: we ignore the non-negativity constraint of the parameters in the optimization problem (3). Thus

$$\min_{\boldsymbol{\beta}\in\mathbb{R}^b}\left[\tfrac{1}{2}\boldsymbol{\beta}^\top\widehat{\boldsymbol{H}}\boldsymbol{\beta} - \widehat{\boldsymbol{h}}^\top\boldsymbol{\beta} + \tfrac{\lambda}{2}\boldsymbol{\beta}^\top\boldsymbol{\beta}\right]. \tag{7}$$

In the above, we included a quadratic regularization term $\lambda\boldsymbol{\beta}^\top\boldsymbol{\beta}/2$, instead of the linear one $\lambda\mathbf{1}_b^\top\boldsymbol{\alpha}$ since the linear penalty term does not work as a regularizer without the non-negativity constraint. Eq.(7) is an unconstrained convex quadratic program, so the solution can be analytically computed. However, since we dropped the non-negativity constraint $\boldsymbol{\beta}\geq\mathbf{0}_b$, some of the learned parameters could be negative. To compensate for this approximation error, we modify the solution by

$$\widehat{\boldsymbol{\beta}} = \max(\mathbf{0}_b, \widetilde{\boldsymbol{\beta}}), \quad \widetilde{\boldsymbol{\beta}} = (\widehat{\boldsymbol{H}} + \lambda\boldsymbol{I}_b)^{-1}\widehat{\boldsymbol{h}}, \tag{8}$$

where $\boldsymbol{I}_b$ is the $b$-dimensional identity matrix and the 'max' operation for vectors is applied in the element-wise manner. This is the solution of the approximation method we propose in this section.

An advantage of the above unconstrained formulation is that the solution can be computed just by solving a system of linear equations. Therefore, the computation is fast and stable. We call this method *unconstrained LSIF* (uLSIF). Due to the $\ell_2$ regularizer, the solution tends to be close to $\mathbf{0}_b$ to some extent. Thus, the effect of ignoring the non-negativity constraint may not be so strong. Below, we theoretically analyze the approximation error of uLSIF.

**Convergence Analysis of uLSIF:** Here, we theoretically analyze the convergence property of the solution $\widehat{\boldsymbol{\beta}}$ of the uLSIF algorithm. Let $\boldsymbol{\beta}^*$ be the optimal solution of the 'ideal' problem: $\boldsymbol{\beta}^* = \max(\mathbf{0}_b, \boldsymbol{\beta}^\circ)$, where $\boldsymbol{\beta}^\circ = \operatorname{argmin}_{\boldsymbol{\beta}\in\mathbb{R}^b}\left[\tfrac{1}{2}\boldsymbol{\beta}^\top\boldsymbol{H}\boldsymbol{\beta} - \boldsymbol{h}^\top\boldsymbol{\beta} + \tfrac{\lambda}{2}\boldsymbol{\beta}^\top\boldsymbol{\beta}\right]$. Then we have

**Theorem 2** *Assume that (a) $\beta_\ell^\circ \neq 0$ for $\ell = 1, \ldots, b$, and (b) $n_{\mathrm{tr}}$ and $n_{\mathrm{te}}$ satisfy $n_{\mathrm{te}} = \omega(n_{\mathrm{tr}}^2)$. Then we have $\mathbb{E}[J(\widehat{\boldsymbol{\beta}})] = J(\boldsymbol{\beta}^*) + \mathcal{O}\left(n_{\mathrm{tr}}^{-1}\right)$.*

Theorem 2 guarantees that uLSIF converges to the ideal solution with order $n_{\mathrm{tr}}^{-1}$. It is possible to explicitly obtain the coefficient of the term of order $n_{\mathrm{tr}}^{-1}$, but we omit the detail due to lack of space.

We can also derive upper bounds on the difference between LSIF and uLSIF and show that uLSIF gives a good approximation to LSIF. However, we do not go into the detail due to space limitation.

**Efficient Computation of Leave-one-out Cross-validation Score:** Another practically very important advantage of uLSIF is that the score of leave-one-out cross-validation (LOOCV) can also be computed analytically—thanks to this property, the computational complexity for performing LOOCV is the same order as just computing a single solution. In the current setting, we are given two sets of samples, $\{\boldsymbol{x}_i^{\mathrm{tr}}\}_{i=1}^{n_{\mathrm{tr}}}$ and $\{\boldsymbol{x}_j^{\mathrm{te}}\}_{j=1}^{n_{\mathrm{te}}}$, which generally have different sample size. For simplicity, we assume that $n_{\mathrm{tr}} < n_{\mathrm{te}}$ and the $i$-th training sample $\boldsymbol{x}_i^{\mathrm{tr}}$ and the $i$-th test sample $\boldsymbol{x}_i^{\mathrm{te}}$ are held out at the same time; the test samples $\{\boldsymbol{x}_j^{\mathrm{te}}\}_{j=n_{\mathrm{tr}}+1}^{n_{\mathrm{te}}}$ are always used for importance estimation.

Let $\widehat{\boldsymbol{\beta}}_\lambda^{(i)}$ be a parameter learned without the $i$-th training sample $\boldsymbol{x}_i^{\mathrm{tr}}$ and the $i$-th test sample $\boldsymbol{x}_i^{\mathrm{te}}$. Then the LOOCV score is expressed as $\frac{1}{n_{\mathrm{tr}}} \sum_{i=1}^{n_{\mathrm{tr}}} [\frac{1}{2}(\boldsymbol{\varphi}(\boldsymbol{x}_i^{\mathrm{tr}})^\top \widehat{\boldsymbol{\beta}}_\lambda^{(i)})^2 - \boldsymbol{\varphi}(\boldsymbol{x}_i^{\mathrm{te}})^\top \widehat{\boldsymbol{\beta}}_\lambda^{(i)}]$. Our approach to efficiently computing the LOOCV score is to use the *Sherman-Woodbury-Morrison formula* for computing matrix inverses—$\widehat{\boldsymbol{\beta}}_\lambda^{(i)}$ can be expressed as $\widehat{\boldsymbol{\beta}}_\lambda^{(i)} = \max\{\boldsymbol{0}_b, \frac{(n_{\mathrm{tr}}-1)n_{\mathrm{te}}}{n_{\mathrm{tr}}(n_{\mathrm{te}}-1)}(\boldsymbol{a} + \frac{\boldsymbol{a}^\top \boldsymbol{\varphi}(\boldsymbol{x}_i^{\mathrm{tr}}) \cdot \boldsymbol{a}_{\mathrm{te}}}{n_{\mathrm{tr}} - \boldsymbol{\varphi}(\boldsymbol{x}_i^{\mathrm{tr}})^\top \boldsymbol{a}_{\mathrm{te}}}) - \frac{(n_{\mathrm{tr}}-1)}{n_{\mathrm{tr}}(n_{\mathrm{te}}-1)}(\boldsymbol{a}_{\mathrm{tr}} + \frac{\boldsymbol{a}_{\mathrm{te}}^\top \boldsymbol{\varphi}(\boldsymbol{x}_i^{\mathrm{tr}}) \cdot \boldsymbol{a}_{\mathrm{tr}}}{n_{\mathrm{tr}} - \boldsymbol{\varphi}(\boldsymbol{x}_i^{\mathrm{tr}})^\top \boldsymbol{a}_{\mathrm{tr}}})\}$, where, $\boldsymbol{a} = \boldsymbol{A}^{-1}\widehat{\boldsymbol{h}}, \boldsymbol{a}_{\mathrm{tr}} = \boldsymbol{A}^{-1}\boldsymbol{\varphi}(\boldsymbol{x}_i^{\mathrm{tr}}), \boldsymbol{a}_{\mathrm{te}} = \boldsymbol{A}^{-1}\boldsymbol{\varphi}(\boldsymbol{x}_i^{\mathrm{te}}), \boldsymbol{A} = \widehat{\boldsymbol{H}} + \frac{(n_{\mathrm{tr}}-1)\lambda}{n_{\mathrm{tr}}}\boldsymbol{I}_b$. This implies that the matrix inverse needs to be computed only once (i.e., $\boldsymbol{A}^{-1}$) for calculating LOOCV scores. Thus LOOCV can be carried out very efficiently without repeating hold-out loops.

## 4 Relation to Existing Methods

*Kernel density estimator* (KDE) is a non-parametric technique to estimate a probability density function. KDE can be used for importance estimation by first estimating $\widehat{p}_{\mathrm{tr}}(\boldsymbol{x})$ and $\widehat{p}_{\mathrm{te}}(\boldsymbol{x})$ separately from $\{\boldsymbol{x}_i^{\mathrm{tr}}\}_{i=1}^{n_{\mathrm{tr}}}$ and $\{\boldsymbol{x}_j^{\mathrm{te}}\}_{j=1}^{n_{\mathrm{te}}}$ and then estimating the importance by $\widehat{w}(\boldsymbol{x}) = \widehat{p}_{\mathrm{te}}(\boldsymbol{x})/\widehat{p}_{\mathrm{tr}}(\boldsymbol{x})$. KDE is efficient in computation since no optimization is involved, and model selection is possible by likelihood cross validation. However, KDE may suffer from the curse of dimensionality.

The *kernel mean matching* (KMM) method allows us to directly obtain an estimate of the importance values at training points without going through density estimation [5]. KMM can overcome the curse of dimensionality by directly estimating the importance using a special property of the Gaussian reproducing kernel Hilbert space. However, there is no objective model selection method for the regularization parameter and kernel width. As for the regularization parameter, we may follow a suggestion in the original paper, which is justified by a theoretical argument to some extent [5]. As for the Gaussian width, we may adopt a popular heuristic to use the median distance between samples, although there seems no strong justification for this. The computation of KMM is rather demanding since a quadratic programming problem has to be solved.

Other approaches to directly estimating the importance is to directly fit an importance model to the true importance—a method based on *logistic regression* (LogReg) [1], or a method based on the kernel model (6) (which is called the *Kullback-Leibler importance estimation procedure*, KLIEP) [9, 6]. Model selection of these methods is possible by cross-validation, which is a significant advantage over KMM. However, LogReg and KLIEP are computationally rather expensive since non-linear optimization problems have to be solved.

The proposed LSIF is qualitatively similar to LogReg and KLIEP, i.e., it can avoid density estimation, model selection is possible, and non-linear optimization is involved. However, LSIF is advantageous over LogReg and KLIEP in that it is equipped with a regularization path tracking algorithm. Thanks to this, model selection of LSIF is computationally much more efficient than LogReg and KLIEP. However, the regularization path tracking algorithm tends to be numerically unstable.

The proposed uLSIF inherits good properties of existing methods such as no density estimation involved and a build-in model selection method equipped. In addition to these preferable properties, the solution of uLSIF can be computed analytically through matrix inversion and therefore uLSIF is computationally very efficient and numerically stable. Furthermore, the closed-form solution of uLSIF allows us to compute the LOOCV score analytically without repeating hold-out loops, which highly contributes to reducing the computation time in the model selection phase.

## 5 Experiments

**Importance Estimation:** Let $p_{\mathrm{tr}}(\boldsymbol{x})$ be the $d$-dimensional normal distribution with mean zero and covariance identity; let $p_{\mathrm{te}}(\boldsymbol{x})$ be the $d$-dimensional normal distribution with mean $(1, 0, \ldots, 0)^\top$ and covariance identity. The task is to estimate the importance at training input points: $\{w(\boldsymbol{x}_i^{\mathrm{tr}})\}_{i=1}^{n_{\mathrm{tr}}}$. We fixed the number of test input points at $n_{\mathrm{te}} = 1000$ and consider the following two settings for the number $n_{\mathrm{tr}}$ of training samples and the input dimension $d$: (a) $n_{\mathrm{tr}} = 100$ and $d = 1, 2, \ldots, 20$, (b) $d = 10$ and $n_{\mathrm{tr}} = 50, 60, \ldots, 150$. We run the experiments 100 times for each $d$, each $n_{\mathrm{tr}}$, and each method, and evaluate the quality of the importance estimates $\{\widehat{w}_i\}_{i=1}^{n_{\mathrm{tr}}}$ by the *normalized mean*

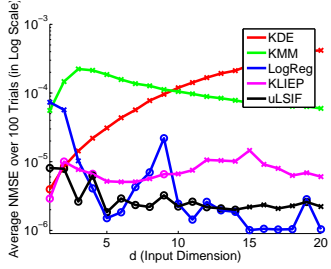

(a) When $d$ is changed

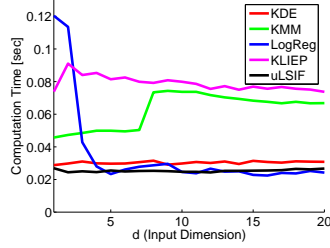

(a) When $d$ is changed

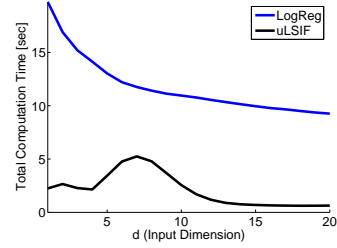

(a) When $d$ is changed

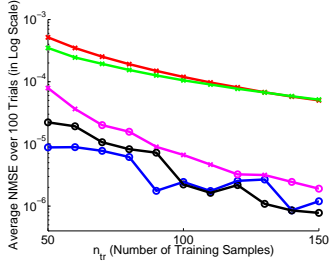

(b) When $n_{\mathrm{tr}}$ is changed

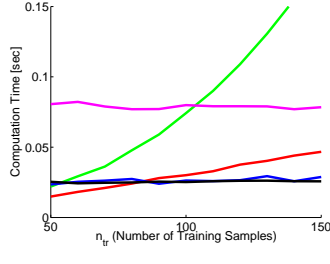

(b) When $n_{\mathrm{tr}}$ is changed

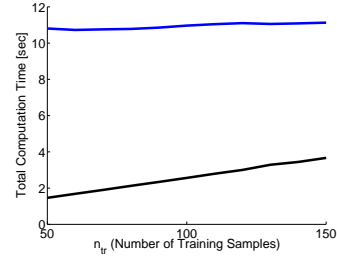

(b) When $n_{\mathrm{tr}}$ is changed

Figure 1: NMSEs averaged over 100 trials in log scale.

Figure 2: Mean computation time (after model selection) over 100 trials.

Figure 3: Mean computation time (including model selection of $\sigma$ and $\lambda$ over $9 \times 9$ grid).

*squared error* (NMSE): $\frac{1}{n_{\mathrm{tr}}} \sum_{i=1}^{n_{\mathrm{tr}}} \left( \widehat{w}(\boldsymbol{x}_i^{\mathrm{tr}}) - w(\boldsymbol{x}_i^{\mathrm{tr}}) \right)^2$, where $\sum_{i=1}^{n_{\mathrm{tr}}} \widehat{w}(\boldsymbol{x}_i^{\mathrm{tr}})$ and $\sum_{i=1}^{n_{\mathrm{tr}}} w(\boldsymbol{x}_i^{\mathrm{tr}})$ are normalized to be one, respectively.

NMSEs averaged over 100 trials (a) as a function of input dimension $d$ and (b) as a function of the training sample size $n_{\mathrm{tr}}$ are plotted in log scale in Figure 1. Error bars are omitted for clear visibility—instead, the best method in terms of the mean error and comparable ones based on the t-test at the significance level 1% are indicated by '∘'; the methods with significant difference are indicated by '×'. Figure 1(a) shows that the error of KDE sharply increases as the input dimension grows, while LogReg, KLIEP, and uLSIF tend to give much smaller errors than KDE. This would be the fruit of directly estimating the importance without going through density estimation. KMM tends to perform poorly, which is caused by an inappropriate choice of the Gaussian kernel width. This implies that the popular heuristic of using the median distance between samples as the Gaussian width is not always appropriate. On the other hand, model selection in LogReg, KLIEP, and uLSIF seems to work quite well. Figure 1(b) shows that the errors of all methods tend to decrease as the number of training samples grows. Again LogReg, KLIEP, and uLSIF tend to give much smaller errors than KDE and KMM.

Next we investigate the computation time. Each method has a different model selection strategy, i.e., KMM does not involve any cross-validation, KDE and KLIEP involve cross-validation over the kernel width, and LogReg and uLSIF involve cross-validation over both the kernel width and the regularization parameter. Thus the naive comparison of the total computation time is not so meaningful. For this reason, we first investigate the computation time of each importance estimation method after the model parameters are fixed. The average CPU computation time over 100 trials are summarized in Figure 2. Figure 2(a) shows that the computation time of KDE, KLIEP, and uLSIF is almost independent of the input dimensionality $d$, while that of KMM and LogReg is rather dependent on $d$. Among them, the proposed uLSIF is one of the fastest methods. Figure 2(b) shows that the computation time of LogReg, KLIEP, and uLSIF is nearly independent of the training sample size $n_{\mathrm{tr}}$, while that of KDE and KMM sharply increase as $n_{\mathrm{tr}}$ increases.

Both LogReg and uLSIF have very good accuracy and their computation time after model selection is comparable. Finally, we compare the entire computation time of LogReg and uLSIF including cross-validation, which is summarized in Figure 3. We note that the Gaussian width $\sigma$ and the regularization parameter $\lambda$ are chosen over the $9 \times 9$ equidistant grid in this experiment for both LogReg and uLSIF. Therefore, the comparison of the entire computation time is fair. Figures 3(a) and 3(b) show that uLSIF is approximately 5 to 10 times faster than LogReg.

Overall, uLSIF is shown to be comparable to the best existing method (LogReg) in terms of the accuracy, but is computationally more efficient than LogReg.

**Covariate Shift Adaptation in Regression and Classification:** Next, we illustrate how the importance estimation methods could be used in *covariate shift adaptation* [8, 5, 1, 9]. Covariate shift is a situation in supervised learning where the input distributions change between the training and test phases but the conditional distribution of outputs given inputs remains unchanged. Under covariate shift, standard learning techniques such as maximum likelihood estimation or cross-validation are biased; the bias caused by covariate shift can be asymptotically canceled by weighting the samples according to the importance. In addition to training input samples $\{\boldsymbol{x}_i^{\mathrm{tr}}\}_{i=1}^{n_{\mathrm{tr}}}$ following a training input density $p_{\mathrm{tr}}(\boldsymbol{x})$ and test input samples $\{\boldsymbol{x}_j^{\mathrm{te}}\}_{j=1}^{n_{\mathrm{te}}}$ following a test input density $p_{\mathrm{te}}(\boldsymbol{x})$, suppose that training *output* samples $\{y_i^{\mathrm{tr}}\}_{i=1}^{n_{\mathrm{tr}}}$ at the training input points $\{\boldsymbol{x}_i^{\mathrm{tr}}\}_{i=1}^{n_{\mathrm{tr}}}$ are given. The task is to predict the outputs for test inputs.

We use the kernel model

$$\widehat{f}(\boldsymbol{x};\boldsymbol{\theta}) = \sum_{\ell=1}^t \theta_\ell K_h(\boldsymbol{x},\boldsymbol{m}_\ell)$$

for function learning, where $K_h(\boldsymbol{x},\boldsymbol{x}')$ is the Gaussian kernel (5) and $\boldsymbol{m}_\ell$ is a template point randomly chosen from $\{\boldsymbol{x}_j^{\mathrm{te}}\}_{j=1}^{n_{\mathrm{te}}}$. We set the number of kernels at $t = 50$. We learn the parameter $\boldsymbol{\theta}$ by *importance weighted regularized least-squares* (IWRLS):

$$\min_{\boldsymbol{\theta}} \left[ \sum_{i=1}^{n_{\mathrm{tr}}} \widehat{w}(\boldsymbol{x}_i^{\mathrm{tr}}) \left( \widehat{f}(\boldsymbol{x}_i^{\mathrm{tr}};\boldsymbol{\theta}) - y_i^{\mathrm{tr}} \right)^2 + \gamma \|\boldsymbol{\theta}\|^2 \right]. \tag{9}$$

It is known that IWRLS is consistent when the true importance $w(\boldsymbol{x}_i^{\mathrm{tr}})$ is used as weights—unweighted RLS is not consistent due to covariate shift, given that the true learning target function $f(\boldsymbol{x})$ is not realizable by the model $\widehat{f}(\boldsymbol{x})$ [8].

The kernel width $h$ and the regularization parameter $\gamma$ in IWRLS (9) are chosen by *importance weighted CV* (IWCV) [9]. More specifically, we first divide the training samples $\{z_i^{\mathrm{tr}} \mid z_i^{\mathrm{tr}} = (\boldsymbol{x}_i^{\mathrm{tr}}, y_i^{\mathrm{tr}})\}_{i=1}^{n_{\mathrm{tr}}}$ into $R$ disjoint subsets $\{\mathcal{Z}_r^{\mathrm{tr}}\}_{r=1}^R$. Then a function $\widehat{f}_r(\boldsymbol{x})$ is learned using $\{\mathcal{Z}_j^{\mathrm{tr}}\}_{j\neq r}$ by IWRLS and its mean test error for the remaining samples $\mathcal{Z}_r^{\mathrm{tr}}$ is computed:

$$\frac{1}{|\mathcal{Z}_r^{\mathrm{tr}}|} \sum_{(\boldsymbol{x},y)\in\mathcal{Z}_r^{\mathrm{tr}}} \widehat{w}(\boldsymbol{x})\mathrm{loss}\left(\widehat{f}_r(\boldsymbol{x}),y\right), \tag{10}$$

where $\mathrm{loss}(\widehat{y},y)$ is $(\widehat{y}-y)^2$ in regression and $\frac{1}{2}(1-\mathrm{sign}\{\widehat{y}y\})$ in classification. We repeat this procedure for $r = 1, \ldots, R$ and choose the kernel width $h$ and the regularization parameter $\gamma$ so that the average of the above mean test error over all $r$ is minimized. We set the number of folds in IWCV at $R = 5$. IWCV is shown to be an (almost) unbiased estimator of the generalization error, while unweighted CV with misspecified models is biased due to covariate shift.

The datasets provided by DELVE and IDA are used for performance evaluation, where training input points are sampled with bias in the same way as [9]. We set the number of samples at $n_{\mathrm{tr}} = 100$ and $n_{\mathrm{te}} = 500$ for all datasets. We compare the performance of KDE, KMM, LogReg, KLIEP, and uLSIF, as well as the uniform weight (Uniform, i.e., no adaptation is made). The experiments are repeated 100 times for each dataset and evaluate the *mean test error*: $\frac{1}{n_{\mathrm{te}}} \sum_{j=1}^{n_{\mathrm{te}}} \mathrm{loss}(\widehat{f}(\boldsymbol{x}_j^{\mathrm{te}}), y_j^{\mathrm{te}})$. The results are summarized in Table 1, where all the error values are normalized by that of the uniform weight (no adaptation). For each dataset, the best method and comparable ones based on the *Wilcoxon signed rank test* at the significance level $1\%$ are described in bold face. The upper half corresponds to regression datasets taken from DELVE while the lower half correspond to classification datasets taken from IDA.

The table shows that the generalization performance of uLSIF tends to be better than that of Uniform, KDE, KMM, and LogReg, while it is comparable to the best existing method (KLIEP). The mean computation time over 100 trials is described in the bottom row of the table, where the value is normalized so that the computation time of uLSIF is one. This shows that uLSIF is computationally more efficient than KLIEP. Thus, proposed uLSIF is overall shown to work well in covariate shift adaptation with low computational cost.

**Outlier Detection:** Here, we consider an outlier detection problem of finding irregular samples in a dataset ("evaluation dataset") based on another dataset ("model dataset") that only contains

Table 1: Covariate shift adaptation. Mean and standard deviation of test error over 100 trials (smaller is better).

| Dataset | Uniform | KDE | KMM | LogReg | KLIEP | uLSIF |
|---|---|---|---|---|---|---|
| kin-8fh | 1.00(0.34) | 1.22(0.52) | 1.55(0.39) | 1.31(0.39) | °0.95(0.31) | °1.02(0.33) |
| kin-8fm | 1.00(0.39) | 1.12(0.57) | 1.84(0.58) | 1.38(0.57) | °0.86(0.35) | °0.88(0.39) |
| kin-8nh | °1.00(0.26) | 1.09(0.20) | 1.19(0.29) | 1.09(0.19) | °0.99(0.22) | °1.02(0.18) |
| kin-8nm | °1.00(0.30) | 1.14(0.26) | 1.20(0.20) | 1.12(0.21) | °0.97(0.25) | 1.04(0.25) |
| abalone | °1.00(0.50) | 1.02(0.41) | °0.91(0.38) | °0.97(0.49) | °0.97(0.69) | °0.96(0.61) |
| image | °1.00(0.51) | 0.98(0.45) | 1.08(0.54) | °0.98(0.46) | °0.94(0.44) | °0.98(0.47) |
| ringnorm | 1.00(0.04) | 0.87(0.04) | °0.87(0.04) | 0.95(0.08) | 0.99(0.06) | 0.91(0.08) |
| twonorm | 1.00(0.58) | 1.16(0.71) | °0.94(0.57) | °0.91(0.61) | °0.91(0.52) | °0.88(0.57) |
| waveform | 1.00(0.45) | 1.05(0.47) | 0.98(0.31) | °0.93(0.32) | °0.93(0.34) | °0.92(0.32) |
| Average | 1.00(0.38) | 1.07(0.40) | 1.17(0.37) | 1.07(0.37) | 0.95(0.35) | 0.96(0.36) |
| Time | — | 0.82 | 3.50 | 3.27 | 3.64 | 1.00 |

Table 2: Outlier detection. Mean AUC values over 20 trials (larger is better).

| Dataset | uLSIF | KLIEP | LogReg | KMM | OSVM | LOF | KDE |
|---|---|---|---|---|---|---|---|
| banana | .851 | .815 | .447 | .578 | .360 | .915 | .934 |
| b-cancer | .463 | .480 | .627 | .576 | .508 | .488 | .400 |
| diabetes | .558 | .615 | .599 | .574 | .563 | .403 | .425 |
| f-solar | .416 | .485 | .438 | .494 | .522 | .441 | .378 |
| german | .574 | .572 | .556 | .529 | .535 | .559 | .561 |
| heart | .659 | .647 | .833 | .623 | .681 | .659 | .638 |
| image | .812 | .828 | .600 | .813 | .540 | .930 | .916 |
| splice | .713 | .748 | .368 | .541 | .737 | .778 | .845 |
| thyroid | .534 | .720 | .745 | .681 | .504 | .111 | .256 |
| titanic | .525 | .534 | .602 | .502 | .456 | .525 | .461 |
| t-norm | .905 | .902 | .161 | .439 | .846 | .889 | .875 |
| w-form | .890 | .881 | .243 | .477 | .861 | .887 | .861 |
| Average | .661 | .685 | .530 | .608 | .596 | .629 | .623 |
| Time | 1.00 | 11.7 | 5.35 | 751 | 12.4 | 85.5 | 8.70 |

regular samples. Defining the importance over two sets of samples, we can see that the importance values for regular samples are close to one, while those for outliers tend to be significantly deviated from one. Thus the importance values could be used as an index of the degree of outlyingness in this scenario. Since the evaluation dataset has wider support than the model dataset, we regard the evaluation dataset as the training set (i.e., the denominator in the importance) and the model dataset as the test set (i.e., the numerator in the importance). Then outliers tend to have smaller importance values (i.e., close to zero).

We again test KMM, LogReg, KLIEP, and uLSIF for importance estimation; in addition, we test native outlier detection methods such as the *one-class support vector machine* (OSVM) [7], the *local outlier factor* (LOF) [3], and the *kernel density estimator* (KDE). The datasets provided by IDA are used for performance evaluation. These datasets are binary classification datasets consisting of training and test samples. We allocate all positive training samples for the "model" set, while all positive test samples and $1\%$ of negative test samples are assigned in the "evaluation" set. Thus, we regard the positive samples as regular and the negative samples as irregular.

The mean AUC values over 20 trials as well as the computation time are summarized in Table 2, showing that uLSIF works fairly well. KLIEP works slightly better than uLSIF, but uLSIF is computationally much more efficient. LogReg overall works rather well, but it performs poorly for some datasets and therefore the average AUC value is small. KMM and OSVM are not comparable to uLSIF both in AUC and computation time. LOF and KDE work reasonably well in terms of AUC, but their computational cost is high. Thus, proposed uLSIF is overall shown to work well and computationally efficient also in outlier detection.

## 6 Conclusions

We proposed a new method for importance estimation that can avoid solving a substantially more difficult task of density estimation. We are currently exploring various possible applications of important estimation methods beyond covariate shift adaptation and outlier detection, e.g., feature selection, conditional distribution estimation, and independent component analysis—we believe that importance estimation could be used as a new versatile tool in machine learning.

## References

[1] S. Bickel et al. Discriminative learning for differing training and test distributions. ICML 2007.

[2] S. Bickel et al. Dirichlet-enhanced spam filtering based on biased samples. NIPS 2006.

[3] M. M. Breunig et al. LOF: Identifying density-based local outliers. SIGMOD 2000.

[4] T. Hastie et al. The entire regularization path for the support vector machine. JMLR 2004.

[5] J. Huang et al. Correcting sample selection bias by unlabeled data. NIPS 2006.

[6] X. Nguyen et al. Estimating divergence functions and the likelihood ratio. NIPS 2007.

[7] B. Schölkopf et al. Estimating the support of a high-dimensional distribution. *Neural Computation*, 13(7):1443–1471, 2001.

[8] H. Shimodaira. Improving predictive inference under covariate shift by weighting the log-likelihood function. *Journal of Statistical Planning and Inference*, 90(2):227–244, 2000.

[9] M. Sugiyama et al. Direct importance estimation with model selection. NIPS 2007.

